# Using Prior Knowledge in a NNPDA to Learn Context-Free Languages

**Sreerupa Das**
Dept. of Comp. Sc. &
Inst. of Cognitive Sc.
University of Colorado
Boulder, CO 80309

**C. Lee Giles***
NEC Research Inst.
4 Independence Way
Princeton, NJ 08540

**Guo-Zheng Sun**
*Inst. for Adv. Comp. Studies
University of Maryland
College Park, MD 20742

## Abstract

Although considerable interest has been shown in language inference and automata induction using recurrent neural networks, success of these models has mostly been limited to regular languages. We have previously demonstrated that Neural Network Pushdown Automaton (NNPDA) model is capable of learning deterministic context-free languages (e.g., $a^n b^n$ and parenthesis languages) from examples. However, the learning task is computationally intensive. In this paper we discus some ways in which *a priori* knowledge about the task and data could be used for efficient learning. We also observe that such knowledge is often an experimental prerequisite for learning nontrivial languages (eg. $a^n b^n c b^m a^m$).

## 1   INTRODUCTION

Language inference and automata induction using recurrent neural networks has gained considerable interest in the recent years. Nevertheless, success of these models has been mostly limited to regular languages. Additional information in form of *a priori* knowledge has proved important and at times necessary for learning complex languages (Abu-Mostafa 1990; Al-Mashouq and Reed, 1991; Omlin and Giles, 1992; Towell, 1990). They have demonstrated that partial information incorporated in a connectionist model guides the learning process through constraints for efficient learning and better generalization.

We have previously shown that the NNPDA model can learn Deterministic Context

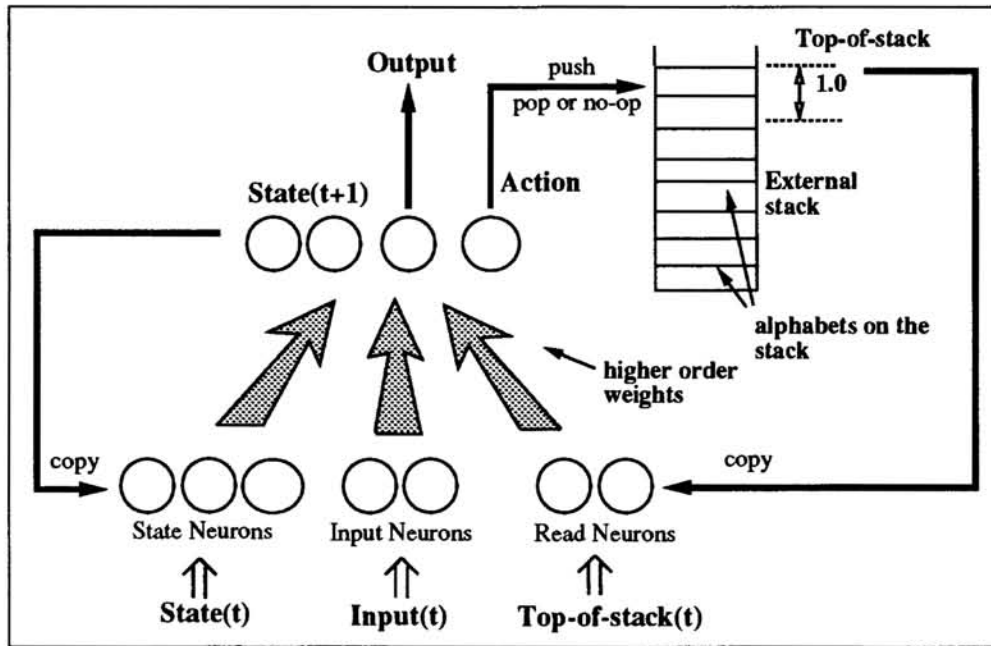

Figure 1: The figure shows the architecture of a third-order NNPDA. Each weight relates the product of Input(t), State(t) and Top-of-Stack information to the State(t+1). Depending on the activation of the Action Neuron, stack action (namely, push, pop or no operation) is taken and the Top-of-Stack (i.e. value of Read Neurons) is updated.

Free Languages (DCFLs) from a finite set of examples. However, the learning task requires considerable amount of time and computational resources. In this paper we discuss methods in which *a priori* knowledge, may be incorporated in a *Neural network Pushdown Automaton (NNPDA)* described in (Das, Giles and Sun, 1992; Giles et al, 1990; Sun et al, 1990).

## 2    THE NEURAL NETWORK PUSHDOWN AUTOMATA

### 2.1    ARCHITECTURE

The description of the network architecture is necessarily brief, for further details see the references above. The network consists of a set of recurrent units, called *state neurons* and an external stack memory. One state neuron is designated as the *output neuron*. The *state neurons* get input (at every time step) from three sources: from their own recurrent connections, from the *input neurons* and from the *read neurons*. The *input neurons* register external inputs which consist of strings of characters presented one at a time. The *read neurons* keep track of the symbol(s) on top of the stack. One non-recurrent state neuron, called the *action neuron*, indicates the stack action (push, pop or no-op) at any instance. The architecture is shown in Figure 1.

The stack used in this model is *continuous*. Unlike an usual discrete stack where an element is either present or absent, elements in a continuous stack may be present in varying degrees (values between [0, 1]). A continuous stack is essential in order

to permit the use of a continuous optimization method during learning. The stack is manipulated by the continuous valued action neuron. A detailed discussion on the operations may be found in (Das, Giles and Sun, 1992).

## 2.2  LEARNABLE CLASS OF LANGUAGES

The class of language learnable by the NNPDA is a proper subset of deterministic context-free languages. A formal description of a Pushdown Automaton (PDA) requires two distinct sets of symbols – one is the input symbol set and the other is the stack symbol set[1]. We have reduced the complexity of this PDA model in the following ways: First, we use the same set of symbols for the input and the stack. Second, when a push operation is performed the symbol pushed on the stack is the one that is available as the current input. Third, no epsilon transitions are allowed in the NNPDA. Epsilon transition is one that performs state transition and stack action without reading in a new input symbol. Unlike a deterministic finite state automata, a deterministic PDA can make epsilon transitions under certain restrictions[1]. Although these simplifications reduce the language class learnable by NNPDA, nevertheless the languages in this class retain essential properties of CFLs and is therefore more complex than any regular language.

## 2.3  TRAINING

The activation of the state neurons s at time step $t+1$ may be formulated as follows (we will only consider third order NNPDA in this paper):

$$s_i(t+1) = g\left(\sum\sum\sum w_{ijkl}s_j(t)i_k(t)r_l(t)\right) \qquad (1)$$

where $g(x) = frac1/1 + exp(-x)$, **i** is the activation of the input neurons and **r** is the activation of the read neuron and **W** is the weight matrix of the network. We use a localized representation for the input and the read symbols. During training, input sequences are presented one at a time and activations are allowed to propagate until the end of the string is reached. Once the end is reached the activation of the output neuron is matched with the *target* (which is 1.0 for positive string and 0.0 for a negative string) The learning rule used in the NNPDA is a significantly enhanced extension to *Real Time Recurrent Learning* (Williams and Zipser, 1989).

## 2.4  OBJECTIVE FUNCTION

The objective function used to train the network consists of two error terms: one for positive strings and the other for negative strings. For positive strings we require (a) the NNPDA must reach a final state and (b) the stack must be empty. This criterion can be reached by minimizing the error function:

$$Error = \frac{1}{2}[(1 - s_o(l))^2 + L(l)^2] \qquad (2)$$

where $S_o(l)$ is the activation of an output neuron and $L(l)$ is the stack length, after a string of length $l$ has been presented as input a character at a time. For negative

| avg of total | parenthesis | | postfix | | $a^n b^n$ | |
|---|---|---|---|---|---|---|
| presentations | w IL | w/o IL | w IL | w/o IL | w IL | w/o IL |
| # of strings | 2671 | 5644 | 8326 | 15912 | 108200 | >200000 |
| # of character | 10628 | 29552 | 31171 | 82002 | 358750 | >700000 |

Table 1: Effect of Incremental Learning (IL) is displayed in this table. The number of strings and characters required for learning the languages are provided here.

| | parenthesis | | $a^n b^n$ | |
|---|---|---|---|---|
| | w SSP | w/o SSP | w SSP | w/o SSP |
| epochs | 50–80 | 50–80 | 150–250 | 150–250 |
| generalization | 100% | 100% | 100% | 98.97% |
| number of units | 1+1 | 2 | 1+1 | 2 |
| | $a^n b^n c b^m a^m$ | | $a^{n+m} b^n c^m$ | |
| | w SSP | w/o SSP | w SSP | w/o SSP |
| epochs | 150 | *** | 150–250 | *** |
| generalization | 96.02% | *** | 100% | *** |
| number of units | 1+1 | *** | 1+1 | *** |

Table 2: This table provides some statistics on epochs, generalization and number of hidden units required for learning with and without selective string presentation (SSP).

strings, the error function is modified as:

$$Error = \begin{cases} s_o(l) - L(l) & \text{if } (s_o(l) - L(l)) > 0.0 \\ 0 & \text{else} \end{cases} \qquad (3)$$

Equation (2) reflects the criterion that, for a negative pattern we require either the final state $s_o(l) = 0.0$ or the stack length $L(l)$ to be greater than 1.0 (only when $s_o(l) = 1.0$ and the stack length $L(l)$ is close to zero, the error is high).

## 3    BUILDING IN PRIOR KNOWLEDGE

In practical inference tasks it may be possible to obtain prior knowledge about the problem domain. In such cases it often helps to build in knowledge into the system under study. There could be at least two different types of knowledge available to a model (a) knowledge that depends on the training data with absolutely no knowledge about the automaton, and (b) partial knowledge about the automaton being inferred. Some of ways in which knowledge can be provided to the model are discussed below.

### 3.1    KNOWLEDGE FROM THE DATA

#### 3.1.1    Incremental Learning

Incremental Learning has been suggested by many (Elman, 1991; Giles et al, 1990, Sun et al, 1990), where the training examples are presented in order of increasing

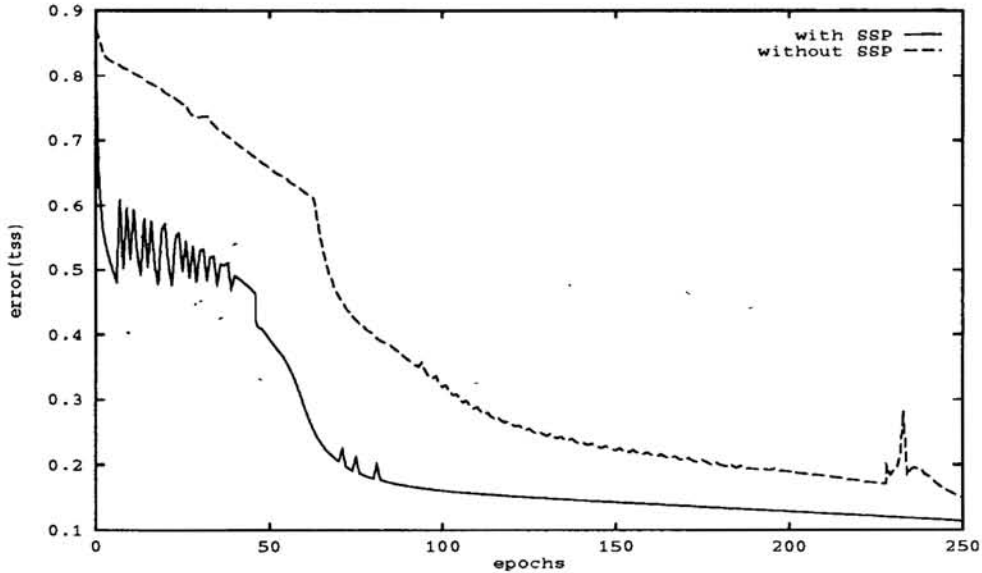

Figure 2: Faster convergence using selective string presentation (SSP) for parenthesis language task.

length. This model of learning starts with a training set containing short simple strings. Longer strings are added to the training set as learning proceeds.

We believe that incremental learning is very useful when (a) the data presented contains structure, and (b) the strings learned earlier embody simpler versions of the task being learned. Both these conditions are valid for context-free languages. Table 1 provides some results obtained when incremental learning was used. The figures are averages over several pairs of simulations, each of which were initialized with the same initial random weights.

### 3.1.2   Selective Input Presentation

Our training data contained both positive and negative examples. One problem with training on incorrect strings is that, once a symbol in the string is reached that makes it negative, no further information is gained by processing the rest of the string. For example, the fifth $a$ in the string $aaaaba...$ makes the string a negative example of the language $a^n b^n$, irrespective of what follows it. In order to incorporate this idea we have introduced the concept of a *dead state*.

During training, we assume that there is a *teacher* or an *oracle* who has knowledge of the grammar and is able to identify the first (leftmost) occurrence of incorrect sequence of symbols in a negative string. When such a point is reached in the input string, further processing of the string is stopped and the network is trained so that one designated state neuron called the *dead state neuron* is active. To accommodate the idea of a *dead state* in the learning rule, the following change is made: if the network is being trained on negative strings that end in a *dead state* then the length $L(l)$ in the error function in equation (1) is ignored and it simply becomes

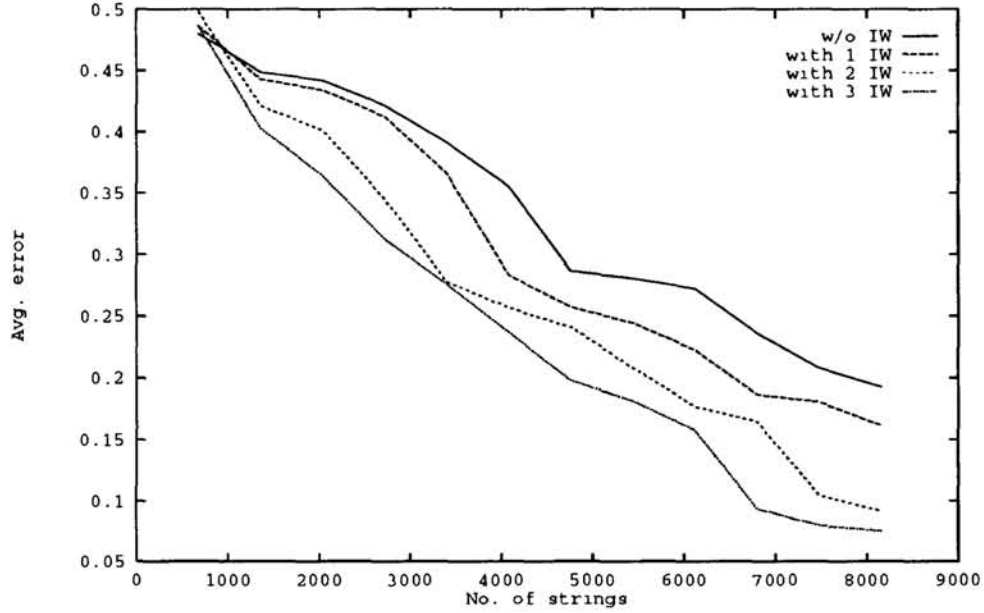

Figure 3: Learning curves when none, one or more initial weights (IW) were set for postfix language learning task

$Error = \frac{1}{2}(1 - S_{dead}(l))^2$. Since such strings have an negative subsequence, they cannot be a prefix to any positive string. Therefore at this point we do not care about the length of the stack. For strings that are either positive or negative but do not go to a dead state (an example would be a prefix of a positive string); the objective function remains the same as described earlier in Equations 1 and 2.

Such additional information provided during training resulted in efficient learning, helped in learning of exact pushdown automata and led to better generalization for the trained network. Information in this form was often a prerequisite for successfully learning certain languages. Figure 2 shows a typical plot of improvement in learning when such knowledge is used. Table 2 shows improvements in the statistics for generalization, number of units needed and number of epochs required for learning. The numbers in the tables were averages over several simulations; changing the initial conditions resulted in values of similar orders of magnitude.

## 3.2   KNOWLEDGE ABOUT THE TASK

### 3.2.1   Knowledge About The Target PDA's Dynamics

One way in which knowledge about the target PDA can be built into a system is by biasing the initial conditions of the network. This may be done by assigning predetermined initial values to a selected set of weights (or biases). For example a third order NNPDA has a dynamics that maps well onto the theoretical model of a PDA. Both allow a three to two mapping of a similar kind. This is because in the third order NNPDA, the product of the activations of the input neurons, the read neurons and the state neurons determine the next state and the next action to be

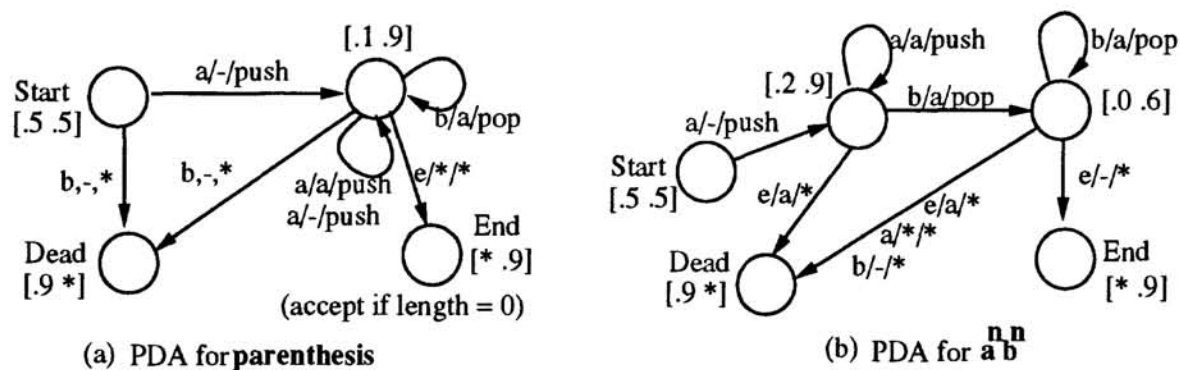

(a) PDA for **parenthesis**                    (b) PDA for $a^n b^n$

Figure 4: The figure shows some of the PDAs inferred by the NNPDA. In the figure the nodes in the graph represent states inferred by the NNPDA and the numbers in "[ ]" indicates the state representations. Every transition is indicated by an arrow and is labeled as "x/y/z" where "x" corresponds to the current input symbol, "y" corresponds to the symbol on top of the stack and "z" corresponds to the action taken.

taken. It may be possible to determine some of the weights in a third order network if certain information about the automaton in known. Typical improvement in learning is shown in Figure 3 for a postfix language learning task.

### 3.2.2    Using Structured Examples

Structured examples from a grammar are a set of strings where the order of letter generation is indicated by brackets. An example would be the string ((ab)c) generated by the rules $S \rightarrow Xc; X \rightarrow ab$. Under the current dynamics and limitations of the model, this information could be interpreted as providing the stack actions (push and pop) to the NNPDA. Learning the palindrome language is a hard task because it necessitates remembering a precise history over a long period of time. The NNPDA was able to learn the palindrome language for two symbols when structured examples were presented.

## 4    AUTOMATON EXTRACTION FROM NNPDA

Once the network performs well on the training set, the transition rules in the inferred PDA can then be deduced. Since the languages learned by the NNPDA so far corresponded to PDAs with few states, the state representations in the induced PDA could be inferred by looking at the state neuron activations when presented with all possible character sequences. For larger PDAs clustering techniques could be used to infer the state representations. Various clustering techniques for similar tasks have been discussed in (Das and Das, 1992; Giles et al., 1992). Figure 4 shows some of the PDAs inferred by the NNPDA.

## 5  CONCLUSION

This paper has described some of the ways in which prior knowledge could be used to learn DCFGs in an NNPDA. Such knowledge is valuable to the learning process in two ways. It may reduce the solution space, and as a consequence may speed up the learning process. Having the right restrictions on a given representation can make learning simple: which reconfirms an old truism in Artificial Intelligence.

## Footnotes

[1]For details refer to (Hopcroft, 1979).

### References

Y.S. Abu-Mostafa. (1990) Learning from hints in neural networks. *Journal of Complexity*, 6:192-198.

K.A. Al-Mashouq and I.S. Reed. (1991) Including hints in training neural networks. *Neural Computation*, 3(3):418-427.

S. Das and R. Das. (1992) Induction of discrete state-machine by stabilizing a continuous recurrent network using clustering. *To appear in CSI Journal of Computer Science and Informatics*. Special Issue on Neural Computing.

S. Das, C.L. Giles, and G.Z. Sun. (1992) Learning context free grammars: capabilities and limitations of neural network with an external stack memory. *Proc of the Fourteenth Annual Conf of the Cognitive Science Society*, pp. 791-795. Morgan Kaufmann, San Mateo, Ca.

J.L. Elman. (1991) Incremental learning, or the importance of starting small. CRL Tech Report 9101, Center for Research in Language, UCSD, La Jolla, CA.

C.L. Giles, G.Z. Sun, H.H. Chen, Y.C. Lee and D. Chen, (1990) Higher Order Recurrent Networks & Grammatical Inference, *Advances in Neural Information Processing Systems 2*, pp. 380-387, ed. D.S. Touretzky, Morgan Kaufmann, San Mateo, CA.

C.L. Giles, C.B. Miller, H.H. Chen, G.Z. Sun, and Y.C. Lee. (1992) Learning and extracting finite state automata with second-order recurrent neural networks. *Neural Computation*, 4(3):393-405.

J.E. Hopfcroft and J.D. Ullman. (1979) *Introduction to Automata Theory, Languages and Computation*. Addison-Wesley, Reading, MA.

C.W. Omlin and C.L. Giles. (1992) Training second-order recurrent neural networks using hints. *Proceedings of the Ninth Int Conf on Machine Learning*, pp. 363-368. D. Sleeman and P. Edwards (eds). Morgan Kaufmann, San Mateo, Ca.

G.Z. Sun, H.H. Chen, C.L. Giles, Y.C. Lee and D. Chen. (1991) Neural networks with external memory stack that learn context-free grammars from examples. *Proc of the Conf on Information Science and Systems*, Princeton U., Vol. II, pp. 649-653.

G.G. Towell, J.W. Shavlik and M.O Noordewier. (1990) Refinement of approximately correct domain theories by knowledge-based neural-networks. In *Proc of the Eighth National Conf on Artificial Intelligence*, Boston, MA. pp. 861.

R.J. Williams and D. Zipser. (1989) A learning algorithm for continually running fully recurrent neural networks. *Neural Computation* 1(2):270-280.
